# The Fidelity of Local Ordinal Encoding

**Javid Sadr, Sayan Mukherjee, Keith Thoresz, Pawan Sinha**
Center for Biological and Computational Learning
Department of Brain and Cognitive Sciences, MIT
Cambridge, Massachusetts, 02142 USA
{*sadr,sayan,thorek,sinha*}*@ai.mit.edu*

## Abstract

A key question in neuroscience is how to encode sensory stimuli such as images and sounds. Motivated by studies of response properties of neurons in the early cortical areas, we propose an encoding scheme that dispenses with absolute measures of signal intensity or contrast and uses, instead, only local ordinal measures. In this scheme, the structure of a signal is represented by a set of equalities and inequalities across adjacent regions. In this paper, we focus on characterizing the fidelity of this representation strategy. We develop a regularization approach for image reconstruction from ordinal measures and thereby demonstrate that the ordinal representation scheme can faithfully encode signal structure. We also present a neurally plausible implementation of this computation that uses only local update rules. The results highlight the robustness and generalization ability of local ordinal encodings for the task of pattern classification.

## 1  Introduction

Biological and artificial recognition systems face the challenge of grouping together differing proximal stimuli arising from the same underlying object. How well the system succeeds in overcoming this challenge is critically dependent on the nature of the internal representations against which the observed inputs are matched. The representation schemes should be capable of efficiently encoding object concepts while being tolerant to their appearance variations.

In this paper, we introduce and characterize a biologically plausible representation scheme for encoding signal structure. The scheme employs a simple vocabulary of local ordinal relations, of the kind that early sensory neurons are capable of extracting. Our results so far suggest that this scheme possesses several desirable characteristics, including tolerance to object appearance variations, computational simplicity, and low memory requirements. We develop and demonstrate our ideas in the visual domain, but they are intended to be applicable to other sensory modalities as well.

The starting point for our proposal lies in studies of the response properties of neurons in the early sensory cortical areas. These response properties constrain

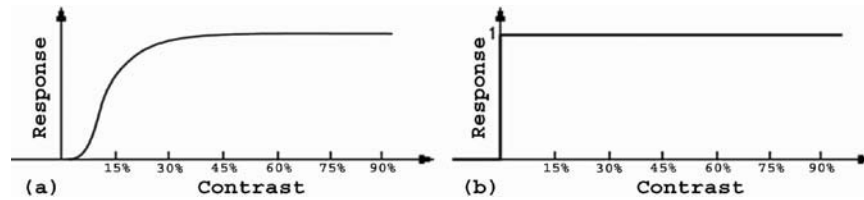

Figure 1: (a) A schematic contrast response curve for a primary visual cortex neuron. The response of the neuron saturates at low contrast values. (b) An idealization of (a). This unit can be thought of as an ordinal comparator, providing information only about contrast polarity but not its magnitude.

the kinds of measurements that can plausibly be included in our representation scheme. In the visual domain, many striate cortical neurons have rapidly saturating contrast response functions [1, 4]. Their tendency to reach ceiling level responses at low contrast values render these neurons sensitive primarily to local ordinal, rather than metric, relations. We propose to use an idealization of such units as the basic vocabulary of our representation scheme (figure 1). In this scheme, objects are encoded as sets of local ordinal relations across image regions. As discussed below, this very simple idea seems well suited to handling the photometric appearance variations that real-world objects exhibit.

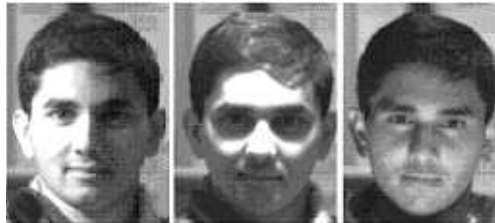

Figure 2: The challenge for a representation scheme: to construct stable descriptions of objects despite radical changes in appearance.

As figure 2 shows, variations in illumination significantly alter the individual brightness of different parts of the face, such as the eyes, cheeks, and forehead. Therefore, absolute image brightness distributions are unlikely to be adequate for classifying all of these images as depicting the same underlying object. Even the contrast magnitudes across different parts of the face change greatly under different lighting conditions. While the absolute luminance and contrast magnitude information is highly variable across these images, Thoresz and Sinha [9] have shown that one can identify some stable ordinal measurements. Figure 3 shows several pairs of average brightness values over localized patches for each of the three images included in figure 2. Certain regularities are apparent. For instance, the average brightness of the left eye is always less than that of the forehead, irrespective of the lighting conditions. The relative magnitudes of the two brightness values may change, but the sign of the inequality does not. In other words, the ordinal relationship between the average brightnesses of the <left-eye, forehead> pair is invariant under lighting changes. Figure 3 shows several other such pair-wise invariances. It seems, therefore that local ordinal relations may encode the stable facial attributes across different illumination conditions. An additional advantage to using ordinal relations is their natural robustness to sensor noise. Thus, it would seem that local ordinal representations may be well suited for devising compact representations, robust against

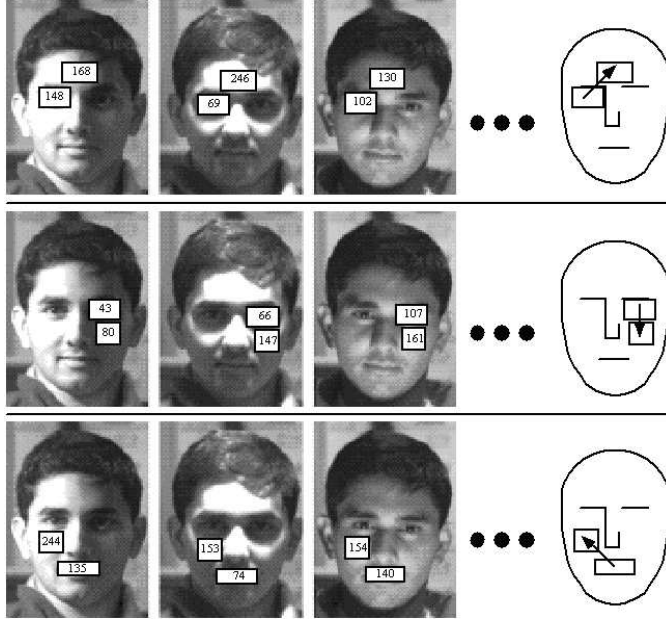

Figure 3: The absolute brightnesses and their relative magnitudes change under different lighting conditions but several pair-wise ordinal relationships stay invariant.

large photometric variations, for at least some classes of objects. Notably, for similar reasons, ordinal measures have also been shown to be a powerful tool for simple, efficient, and robust stereo image matching [3].

In what follows, we address an important open question regarding the expressiveness of the ordinal representation scheme. Given that this scheme ignores absolute luminance and contrast magnitude information, an obvious question that arises is whether such a crude representation strategy can encode object/image structure with any fidelity.

## 2   Information Content of Local Ordinal Encoding

Figure 4 shows how we define ordinal relations between an image region $p_a$ and its immediate neighbors $p_b = \{p_{a1}, \ldots, p_{a8}\}$. In the conventional rectilinear grid, when all image regions $p_a$ are considered, four of the eight relations are redundant; we encode the remaining four as $\{1, 0, -1\}$ based on the difference in luminance between two neighbors being positive, zero, or negative, respectively. To demonstrate the richness of information encoded by this scheme, we compare the original image to one produced by a function that reconstructs the image using local ordinal relationships as constraints. Our reconstruction function has the form

$$f(\mathbf{x}) = \mathbf{w} \cdot \phi(\mathbf{x}), \tag{1}$$

where $\mathbf{x} = \{i, j\}$ is the position of a pixel, $f(\mathbf{x})$ is its intensity, $\phi$ is a map from the input space into a high (possibly infinite) dimensional space, $\mathbf{w}$ is a hyperplane in this high-dimensional space, and $\mathbf{u} \cdot \mathbf{v}$ denotes an inner product.

Infinitely many reconstruction functions could satisfy the given ordinal constraints. To make the problem well-posed we regularize [10] the reconstruction function subject to the ordinal constraints, as done in ordinal regression for ranking document

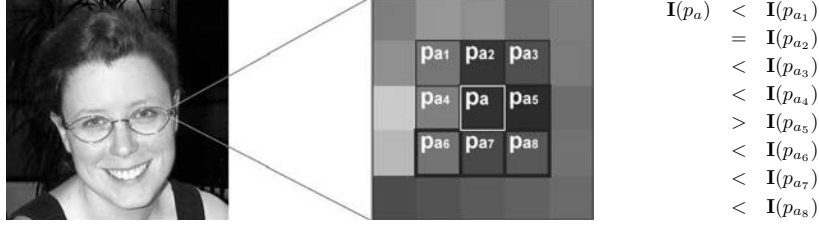

Figure 4: Ordinal relationships between an image region $p_a$ and its neighbors.

retrieval results [5]. Our regularization term is a norm in a Reproducing Kernel Hilbert Space (RKHS) [2, 11]. Minimizing the norm in a RKHS subject to the ordinal constraints corresponds to the following convex constrained quadratic optimization problem:

$$\min_{\xi, \mathbf{w}} \frac{1}{2} ||\mathbf{w}||^2 + C \sum_p \xi_p \qquad (2)$$

subject to

$$\theta(\delta_p) \mathbf{w} \cdot (\phi(\mathbf{x}_{p_a}) - \phi(\mathbf{x}_{p_b})) \geq |\delta_p| - \xi_p, \quad \forall p \text{ and } \xi \geq 0, \qquad (3)$$

where the function $\theta(y) = +1$ for $y \geq 0$ and $-1$ otherwise, $p$ is the index over all pairwise ordinal relations between all pixels $p_a$ and their local neighbors $p_b$ (as depicted in figure 4), $\xi_p$ are slack variables which are penalized by $C$ (the trade-off between smoothness and ordinal constraints), and $\delta_p$ take integer values $\{-1, 0, 1\}$ denoting the ordinal relation (less than, equal to, or greater than, respectively) between $p_a$ and $p_b$; for the case $\delta_p = 0$ the inequality in (3) will be a strict equality.

Taking the dual of (2) subject to constraints (3) results in the following convex quadratic optimization problem which has only box constraints:

$$\max_\alpha \sum_p |\delta_p| \alpha_p - \frac{1}{2} \sum_p \sum_q \alpha_p \alpha_q \tilde{\mathbf{K}}_{pq} \qquad (4)$$

subject to

$$\begin{array}{llll} 0 \leq \alpha_p \leq C & \text{if} & \delta_p > 0, \\ -C \leq \alpha_p \leq C & \text{if} & \delta_p = 0, \\ -C \leq \alpha_p \leq 0 & \text{if} & \delta_p < 0, \end{array} \qquad (5)$$

where $\alpha_p$ are the dual Lagrange multipliers, and the elements of the matrix $\tilde{\mathbf{K}}$ have the form

$$\begin{aligned} \tilde{K}_{pq} &= (\phi(\mathbf{x}_{p_a}) - \phi(\mathbf{x}_{p_b})) \cdot (\phi(\mathbf{x}_{q_a}) - \phi(\mathbf{x}_{q_b})) \\ &= K(\mathbf{x}_{p_a}, \mathbf{x}_{q_a}) - K(\mathbf{x}_{p_b}, \mathbf{x}_{q_a}) - K(\mathbf{x}_{p_a}, \mathbf{x}_{q_b}) + K(\mathbf{x}_{p_b}, \mathbf{x}_{q_b}), \end{aligned}$$

where $K(\mathbf{y}, \mathbf{x}) = \phi(\mathbf{y}) \cdot \phi(\mathbf{x})$ using the standard kernel trick [8]. In this paper we use only Gaussian kernels $K(\mathbf{y}, \mathbf{x}) = \exp(-||\mathbf{x} - \mathbf{y}||^2 / 2\sigma^2)$. The reconstruction function, $f(\mathbf{x})$, obtained from optimizing (4) subject to box constraints (5) has the following form

$$f(\mathbf{x}) = \sum_p \alpha_p \left( K(\mathbf{x}, \mathbf{x}_{p_a}) - K(\mathbf{x}, \mathbf{x}_{p_b}) \right). \qquad (6)$$

Note that in general many of the $\alpha_p$ values may be zero – these terms do not contribute to the reconstruction, and the corresponding constraints in (3) were not

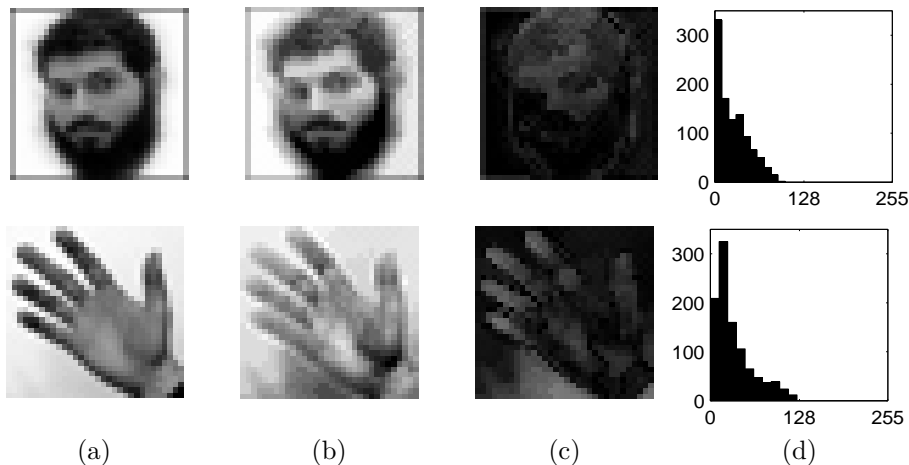

| (a) | (b) | (c) | (d) |

Figure 5: Reconstruction results from the regularization approach. (a) Original images. (b) Reconstructed images. (c) Absolute difference between original and reconstruction. (d) Histogram of absolute difference.

required. The remaining $\alpha_p$ with absolute value less than $C$ satisfy the inequality constraints in (3), whereas those with absolute value at $C$ violate them.

Figure 5 depicts two typical reconstructions performed by this algorithm. The difference images and error histograms suggests that the reconstructions closely match the source images.

## 3   Discussion

Our reconstruction results suggest that the local ordinal representation can faithfully encode image structure. Thus, even though individual ordinal relations are insensitive to absolute luminance or contrast magnitude, a set of such relations implicitly encodes metric information. In the context of the human visual system, this result suggests that the rapidly saturating contrast response functions of the early visual neurons do not significantly hinder their ability to convey accurate image information to subsequent cortical stages.

An important question that arises here is what are the strengths and limitations of local ordinal encoding. The first key limitation is that for any choice of neighborhood size over which ordinal relations are extracted, there are classes of images for which the local ordinal representation will be unable to encode the metric structure. For a neighborhood of size $n$, an image with regions of different luminance embedded in a uniform background and mutually separated by a distance greater than $n$ would constitute such an image. In general, sparse images present a problem for this representation scheme, as might foveal or cortical "magnification," for example. This issue could be addressed by using ordinal relations across multiple scales, perhaps in an adaptive way that varies with the smoothness or sparseness of the stimulus.

Second, the regularization approach above seems biologically implausible. Our intent in using this approach for reconstructions was to show via well-understood theoretical tools the richness of information that local ordinal representations pro-

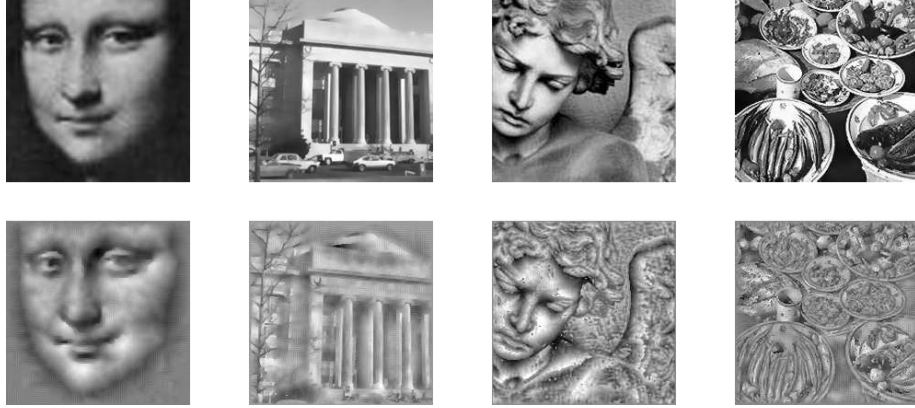

Figure 6: Reconstruction results from the relaxation approach.

vide. In order to address the neural plausibility requirement, we have developed a simple relaxation-based approach with purely local update rules of the kind that can easily be implemented by cortical circuitry. Each unit communicates only with its immediate neighbors and modifies its value incrementally up or down (starting from an arbitrary state) depending on the number of ordinal relations in the positive or negative direction. This computation is performed iteratively until the network settles to an equilibrium state. The update rule can be formally stated as

$$\mathbf{R}_{p_a,t+1} = \mathbf{R}_{p_a,t} + \Delta \sum_{p_b}(\theta(\mathbf{R}_{p_a,t} - \mathbf{R}_{p_b,t}) - \theta(\mathbf{I}_{p_a} - \mathbf{I}_{p_b})), \qquad (7)$$

where $\mathbf{R}_{p_a,t}$ is the intensity of the reconstructed pixel $p_a$ at step $t$, $\mathbf{I}_{p_a}$ is the intensity of the corresponding pixel in the original image, $\Delta$ is a positive update rate, and $\theta$ and $p_b$ are as described above. Figure 6 shows four examples of image reconstructions performed using a relaxation-based approach.

A third potential limitation is that the scheme does not appear to constitute a compact code. If each pixel must be encoded in terms of its relations with all of its eight neighbors, where each relation takes one of three values, $\{-1, 0, 1\}$, then what has been gained over the original image where each pixel is encoded by 8 bits? There are three ways to address this question.

1. Eight relations per pixel is highly redundant – four are sufficient. In fact, as shown in figure 7, the scheme can also tolerate several missing relations.

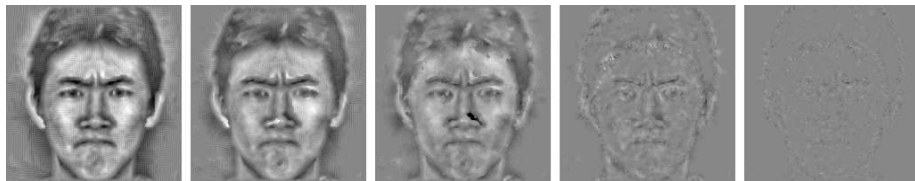

Figure 7: Five reconstructions, shown here to demonstrate the robustness of local ordinal encoding to missing inputs. From left to right: reconstructions based on 100%, 80%, 60%, 40%, and 20% of the full set of immediate neighbor relations.

2. An advantage to using ordinal relations is that they can be extracted and transmitted much more reliably than metric ones. These relations share the same spirit

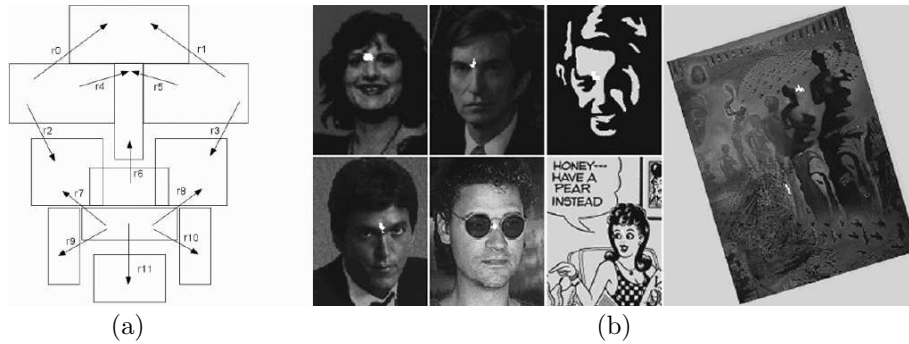

(a)            (b)

Figure 8: A small collection of ordinal relations (a), though insufficient for high fidelity reconstruction, is very effective for pattern classification despite significant appearance variations. (b) Results of using a local ordinal relationship based template to detect face patterns. The program places white dots at the centers of patches classified as faces. (From Thoresz and Sinha, in preparation.)

as loss functions used in robust statistics [6] and trimmed or Winsorized estimators.

3. The intent of the visual system is often not to encode/reconstruct images with perfect fidelity, but rather to encode the most stable characteristics that can aid in classification. In this context, a few ordinal relations may suffice for encoding objects reliably. Figure 8 shows the results of using less than 20 relations for detecting faces. Clearly, such a small set would not be sufficient for reconstructions, but it works well for classification. Its generalization arises because it defines an equivalence class of patterns.

In summary, the ordinal representation scheme provides a neurally plausible strategy for encoding signal structure. While in this paper we focus on demonstrating the fidelity of this scheme, we believe that its true strength lies in defining equivalence classes of patterns enabling generalizations over appearance variations in objects. Several interesting directions remain to be explored. These include the study of ordinal representations across multiple scales, learning schemes for identifying subsets of ordinal relations consistent across different instances of an object, and the relationship of this work to multi-dimensional scaling [12] and to the use of truncated, quantized wavelet coefficients as "signatures" for fast, multiresolution image querying [7].

## Acknowledgements

We would like to thank Gadi Geiger, Antonio Torralba, Ryan Rifkin, Gonzalo Ramos, and Tabitha Spagnolo. Javid Sadr is a Howard Hughes Medical Institute Pre-Doctoral Fellow.

## References

[1] A. Anzai, M. A. Bearse, R. D. Freeman, and D. Cai. Contrast coding by cells in the cat's striate cortex: monocular vs. binocular detection. *Visual Neuroscience*, 12:77–93, 1995.

[2] N. Aronszajn. Theory of reproducing kernels. *Trans. Amer. Math. Soc.*, 686:337–404, 1950.

[3] D. Bhat and S. Nayar. Ordinal measures for image correspondence. In *IEEE Conf. on Computer Vision and Pattern Recognition*, pages 351–357, 1996.

[4] G. C. DeAngelis, I. Ohzawa, and R. D. Freeman. Spatiotemporal organization of simple-cell receptive fields in the cat's striate cortex. i. general characteristics and postnatal development. *J. Neurophysiology*, 69:1091–1117, 1993.

[5] R. Herbrich, T. Graepel, and K. Obermeyer. Support vector learning for ordinal regression. In *Proc. of the Ninth Intl. Conf. on Artificial Neural Networks*, pages 97–102, 1999.

[6] P. Huber. *Robust Statistics*. John Wiley and Sons, New York, 1981.

[7] C. E. Jacobs, A. Finkelstein, and D. H. Salesin. Fast multiresolution image querying. In *Computer Graphics Proc., Annual Conf. Series (SIGGRAPH 95)*, pages 277–286, 1995.

[8] T. Poggio. On optimal nonlinear associative recall. *Biological Cybernetics*, 19:201–209, 1975.

[9] K. Thoresz and P. Sinha. Qualitative representations for recognition. *Vision Sciences Society Abstracts*, 1:81, 2001.

[10] A. N. Tikhonov and V. Y. Arsenin. *Solutions of Ill-posed Problems*. W. H. Winston, Washington, D.C., 1977.

[11] G. Wahba. *Spline Models for Observational Data*. Series in Applied Mathematics, Vol 59, SIAM, Philadelphia, 1990.

[12] F. W. Young and C. H. Null. Mds of nominal data: the recovery of metric information with alscal. *Psychometika*, 53.3:367–379, 1978.
